# Probabilistic Inference of Alternative Splicing Events in Microarray Data

**Ofer Shai, Brendan J. Frey, and Quaid D. Morris**
Dept. of Electrical & Computer Engineering
University of Toronto, Toronto, ON

**Qun Pan, Christine Misquitta, and Benjamin J. Blencowe**
Banting & Best Dept. of Medical Research
University of Toronto, Toronto, ON

## Abstract

Alternative splicing (AS) is an important and frequent step in mammalian gene expression that allows a single gene to specify multiple products, and is crucial for the regulation of fundamental biological processes. The extent of AS regulation, and the mechanisms involved, are not well understood. We have developed a custom DNA microarray platform for surveying AS levels on a large scale. We present here a generative model for the AS Array Platform (GenASAP) and demonstrate its utility for quantifying AS levels in different mouse tissues. Learning is performed using a variational expectation maximization algorithm, and the parameters are shown to correctly capture expected AS trends. A comparison of the results obtained with a well-established but low through-put experimental method demonstrate that AS levels obtained from GenASAP are highly predictive of AS levels in mammalian tissues.

## 1 Biological diversity through alternative splicing

Current estimates place the number of genes in the human genome at approximately 30,000, which is a surprisingly small number when one considers that the genome of yeast, a single-celled organism, has 6,000 genes. The number of genes alone cannot account for the complexity and cell specialization exhibited by higher eukaryotes (i.e. mammals, plants, etc.). Some of that added complexity can be achieved through the use of alternative splicing, whereby a single gene can be used to code for a multitude of products.

Genes are segments of the double stranded DNA that contain the information required by the cell for protein synthesis. That information is coded using an alphabet of 4 (A, C, G, and T), corresponding to the four nucleotides that make up the DNA. In what is known as the *central dogma of molecular biology*, DNA is transcribed to RNA, which in turn is translated into proteins. Messenger RNA (mRNA) is synthesized in the nucleus of the cell and carries the genomic information to the ribosome. In eukaryotes, genes are generally comprised of both *exons*, which contain the information needed by the cell to synthesize proteins, and *introns*, sometimes referred to as spacer DNA, which are spliced out of the pre-mRNA to create mature mRNA. An estimated 35%-75% of human genes [1] can be

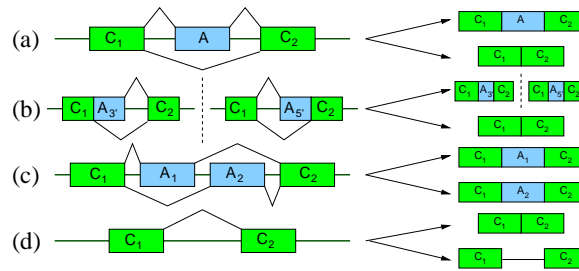

Figure 1: Four types of AS. Boxes represent exons and lines represent introns, with the possible splicing alternatives indicated by the connectors. (a) Single cassette exon inclusion/exclusion. $C_1$ and $C_2$ are constitutive exons (exons that are included in all isoforms) and flank a single alternative exon (A). The alternative exon is included in one isoform and excluded in the other. (b) Alternative 3' (or donor) and alternative 5' (acceptor) splicing sites. Both exons are constitutive, but may contain alternative donor and/or acceptor splicing sites. (c) Mutually exclusive exons. One of the two alternative exons ($A_1$ and $A_2$) may be included in the isoform, but not both. (d) Intron inclusion. An intron may be included in the mature mRNA strand.

spliced to yield different combinations of exons (called *isoforms*), a phenomenon referred to as *alternative splicing* (AS). There are four major types of AS as shown in Figure 1. Many multi-exon genes may undergo more than one alternative splicing event, resulting in many possible isoforms from a single gene. [2]

In addition to adding to the genetic repertoire of an organism by enabling a single gene to code for more than one protein, AS has been shown to be critical for gene regulation, contributing to tissue specificity, and facilitating evolutionary processes. Despite the evident importance of AS, its regulation and impact on specific genes remains poorly understood. The work presented here is concerned with the inference of single cassette exon AS levels (Figure 1a) based on data obtained from RNA expression arrays, also known as microarrays.

## 1.1 An exon microarray data set that probes alternative splicing events

Although it is possible to directly analyze the proteins synthesized by a cell, it is easier, and often more informative, to instead measure the abundance of mRNA present. Traditionally, gene expression (abundance of mRNA) has been studied using low throughput techniques (such as RT-PCR or Northern blots), limited to studying a few sequences at a time and making large scale analysis nearly impossible.

In the early 1990s, microarray technology emerged as a method capable of measuring the expression of thousands of DNA sequences simultaneously. Sequences of interest are deposited on a substrate the size of a small microscope slide, to form probes. The mRNA is extracted from the cell and reverse-transcribed back into DNA, which is labelled with red and green fluorescent dye molecules (cy3 and cy5 respectively). When the sample of tagged DNA is washed over the slide, complementary strands of DNA from the sample hybridize to the probes on the array forming A-T and C-G pairings. The slide is then scanned and the fluorescent intensity is measured at each probe. It is generally assumed that the intensity measure at the probe is linearly related to the abundance of mRNA in the cell over a wide dynamic range.

Despite significant improvements in microarray technologies in recent years, microarray data still presents some difficulties in analysis. Low measurements tend to have extremely low signal to noise ratio (SNR) [7] and probes often bind to sequences that are very similar, but not identical, to the one for which they were designed (a process referred to as cross-

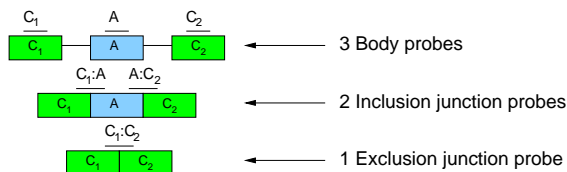

Figure 2: Each alternative splicing event is studied using six probes. Probes were chosen to measure the expression levels of each of the three exons involved in the event. Additionally, 3 probes are used that target the junctions that are formed by each of the two isoforms. The inclusion isoform would express the junctions formed by $C_1$ and A, and A and $C_2$, while the exclusion isoform would express the junction formed by $C_1$ and $C_2$

hybridization). Additionally, probes exhibit somewhat varying hybridization efficiency, and sequences exhibit varying labelling efficiency.

To design our data sets, we mined public sequence databases and identified exons that were strong candidates for exhibiting AS (the details of that analysis are provided elsewhere [4, 3]). Of the candidates, 3,126 potential AS events in 2,647 unique mouse genes were selected for the design of Agilent Custom Oligonucleotide microarray. The arrays were hybridized with unamplified mRNA samples extracted from 10 wild-type mouse tissues (brain, heart, intestine, kidney, liver, lung, salivary gland, skeletal muscle, spleen, and testis). Each AS event has six target probes on the arrays, chosen from regions of the $C_1$ exon, $C_2$ exon, A exon, $C_1$:A splice junction, A:$C_2$ splice junction, and $C_1$:$C_2$ splice junction, as shown in Figure 2.

## 2 Unsupervised discovery of alternative splicing

With the exception of the probe measuring the alternative exon, A (Figure 2), all probes measure sequences that occur in both isoforms. For example, while the sequence of the probe measuring the junction A:$C_1$ is designed to measure the inclusion isoform, half of it corresponds to a sequence that is found in the exclusion isoform. We can therefore safely assume that the measured intensity at each probe is a result of a certain amount of both isoforms binding to the probe. Due to the generally assumed linear relationship between the abundance of mRNA hybridized at a probe and the fluorescent intensity measured, we model the measured intensity as a weighted sum of the overall abundance of the two isoforms.

A stronger assumption is that of a single, consistent hybridization profile for both isoforms across all probes and all slides. Ideally, one would prefer to estimate an individual hybridization profile for each AS event studied across all slides. However, in our current setup, the number of tissues is small (10), resulting in two difficulties. First, the number of parameters is very large when compared to the number of data point using this model, and second, a portion of the events do not exhibit tissue specific alternative splicing within our small set of tissues. While the first hurdle could be accounted for using Baysian parameter estimation, the second cannot.

### 2.1 GenASAP - a generative model for alternative splicing array platform

Using the setup described above, the expression vector $\mathbf{x}$, containing the six microarray measurements as real numbers, can be decomposed as a linear combination of the abundance of the two splice isoforms, represented by the real vector $\mathbf{s}$, with some added noise: $\mathbf{x} = \Lambda\mathbf{s} + noise$, where $\Lambda$ is a $6 \times 2$ weight matrix containing the hybridization profiles for

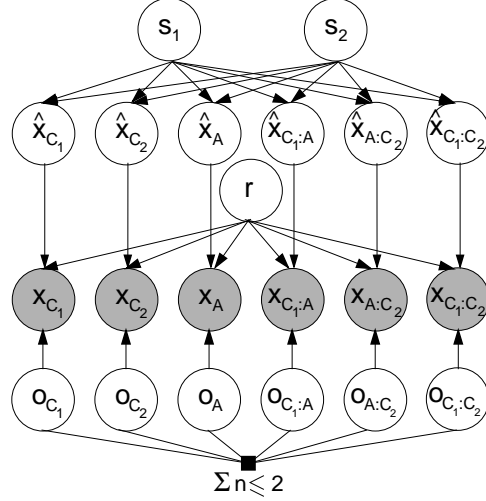

Figure 3: Graphical model for alternative splicing. Each measurement in the observed expression profile, $\mathbf{x}$, is generated by either using a scale factor, $r$, on a linear combination of the isoforms, $\mathbf{s}$, or drawing randomly from an outlier model. For a detailed description of the model, see text.

the two isoforms across the six probes. Note that we may not have a negative amount of a given isoform, nor can the presence of an isoform deduct from the measured expression, and so both $\mathbf{s}$ and $\Lambda$ are constrained to be positive.

Expression levels measured by microarrays have previously been modelled as having expression-dependent noise [7]. To address this, we rewrite the above formulation as

$$\mathbf{x} = r(\Lambda \mathbf{s} + \varepsilon), \tag{1}$$

where $r$ is a scale factor and $\varepsilon$ is a zero-mean normally distributed random variable with a diagonal covariance matrix, $\Psi$, denoted as $p(\varepsilon) = \mathcal{N}(\varepsilon; 0, \Psi)$. The prior distribution for the abundance of the splice isoforms is given by a truncated normal distribution, denoted as $p(\mathbf{s}) \propto \mathcal{N}(\mathbf{s}, 0, I)[\mathbf{s} \geq 0]$, where $[\cdot]$ is an indicator function such that $[\mathbf{s} \geq 0] = 1$ if $\forall i, s_i \geq 0$, and $[\mathbf{s} \geq 0] = 0$ otherwise.

Lastly, there is a need to account for aberrant observations (e.g. due to faulty probes, flakes of dust, etc.) with an outlier model. The complete GenASAP model (shown in Figure 3) accounts for the observations as the outcome of either applying equation (1) or an outlier model. To avoid degenerate cases and ensure meaningful and interpretable results, the number of faulty probes considered for each AS event may not exceed two, as indicated by the filled-in square constraint node in Figure 3.

The distribution of $\mathbf{x}$ conditional on the latent variables, $\mathbf{s}$, $r$, and $\mathbf{o}$, is:

$$p(\mathbf{x}|\mathbf{s}, r, \mathbf{o}) = \prod_i \mathcal{N}(x_i; r\Lambda_i \mathbf{s}, r^2 \Psi_i)^{[o_i=0]} \mathcal{N}(x_i; \mathcal{E}_i, \mathcal{V}_i)^{[o_i=1]}, \tag{2}$$

where $o_i \in \{0, 1\}$ is a bernoulli random variable indicating if the measurement at probe $x_i$ is the result of the AS model or the outlier model parameterized by $p(o_i = 1) = \gamma_i$. The parameters of the outlier model, $\mathcal{E}$ and $\mathcal{V}$, are not optimized and are set to the mean and variance of the data.

## 2.2 Variational learning in the GenASAP model

To infer the posterior distribution over the splice isoform abundances while at the same time learning the model parameters we use a variational expectation-maximization algorithm (EM). EM maximizes the log likelihood of the data by iteratively estimating the posterior distribution of the model given the data in the expectation (E) step, and maximizing the log likelihood with respect to the parameters, while keeping the posterior fixed, in the maximization (M) step. Variational EM is used when, as in the case of GenASAP, the exact posterior is intractable. Variational EM minimizes the free energy of the model, defined as the KL-divergence between the joint distribution of the latent and observed variables and the approximation to the posterior under the model parameters [5, 6].

We approximate the true posterior using the $Q$ distribution given by

$$
Q(\{\mathbf{s}^{(t)}\}, \{\mathbf{o}^{(t)}\}, \{r^{(t)}\}) = \prod_{t=1}^{T} Q(r^{(t)})Q(\mathbf{o}^{(t)}|r^{(t)}) \prod_{i} Q(s_i^{(t)}|o_i^{(t)}, r^{(t)})
$$

$$
= Z^{(t)^{-1}} \prod_{t=1}^{T} \rho^{(t)}\omega^{(t)}\mathcal{N}(\mathbf{s}^{(t)}; \mu_{r\mathbf{o}}^{(t)^d}, \Sigma_{r\mathbf{o}}^{(t)^d})[\mathbf{s}^{(t)} \geq 0],
$$

(3)

where $Z$ is a normalization constant, the superscript $d$ indicates that $\Sigma$ is constrained to be diagonal, and there are $T$ iid AS events. For computational efficiency, $r$ is selected from a finite set, $r \in \{r_1, r_2, \ldots, r_C\}$ with uniform probability. The variational free energy is given by

$$
\mathcal{F}(Q, P) = \sum_{r} \sum_{\mathbf{o}} \int_{\mathbf{s}} Q(\{\mathbf{s}^{(t)}\}, \{\mathbf{o}^{(t)}\}, \{r^{(t)}\}) \log \frac{Q(\{\mathbf{s}^{(t)}\}, \{\mathbf{o}^{(t)}\}, \{r^{(t)}\})}{P(\{\mathbf{s}^{(t)}\}, \{\mathbf{o}^{(t)}\}, \{r^{(t)}\}, \{\mathbf{x}^{(t)}\})}.
$$

(4)

Variational EM minimizes the free energy by iteratively updating the $Q$ distribution's variational parameters ($\rho^{(t)}$, $\omega^{(t)}$, $\mu_{r\mathbf{o}}^{(t)^d}$, and $\Sigma_{r\mathbf{o}}^{(t)^d}$) in the E-step, and the model parameters ($\Lambda$, $\Psi$, $\{r_1, r_2, \ldots, r_C\}$, and $\gamma$) in the M-step. The resulting updates are too long to be shown in the context of this paper and are discussed in detail elsewhere [3]. A few particular points regarding the E-step are worth covering in detail here.

If the prior on $\mathbf{s}$ was a full normal distribution, there would be no need for a variational approach, and exact EM is possible. For a truncated normal distribution, however, the mixing proportions, $Q(r)Q(\mathbf{o}|r)$ cannot be calculated analytically except for the case where $s$ is scalar, necessitating the diagonality constraint. Note that if $\Sigma$ was allowed to be a full covariance matrix, equation (3) would be the true posterior, and we could find the sufficient statistics of $Q(\mathbf{s}^{(t)}|\mathbf{o}^{(t)}, r^{(t)})$:

$$
\mu_{r\mathbf{o}}^{(t)} = (I + \Lambda^T(I - O^{(t)})^T\Psi^{-1}(I - O^{(t)})\Lambda)^{-1}\Lambda^T(I - O^{(t)})^T\Psi^{-1}\mathbf{x}^{(t)}r^{(t)^{-1}}
$$

(5)

$$
\mathbf{\Sigma}_{r\mathbf{o}}^{(t)^{-1}} = (I + \Lambda^T(I - O^{(t)})^T\Psi^{-1}(I - O^{(t)})\Lambda)
$$

(6)

where $O$ is a diagonal matrix with elements $O_{i,i} = o_i$. Furthermore, it can be easily shown that the optimal settings for $\mu^d$ and $\Sigma^d$ approximating a normal distribution with full covariance $\Sigma$ and mean $\mu$ is

$$
\mu_{optimal}^d = \mu
$$

(7)

$$
\Sigma_{optimal}^{d^{-1}} = diag(\Sigma^{-1})
$$

(8)

In the truncated case, equation (8) is still true. Equation (7) does not hold, though, and $\mu_{optimal}^d$ cannot be found analytically. In our experiments, we found that using equation (7) still decreases the free energy every E-step, and it is significantly more efficient than using, for example, a gradient decent method to compute the optimal $\mu^d$.

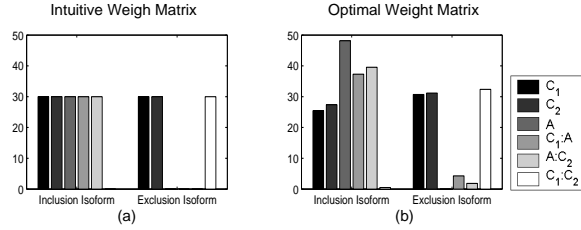

Figure 4: (a) An intuitive set of weights. Based on the biological background, one would expect to see the inclusion isoform hybridize to the probes measuring $C_1$, $C_2$, A, $C_1$:A, and A:$C_2$, while the exclusion isoform hybridizes to $C_1$, $C_2$, and $C_1$:$C_2$. (b) The learned set of weights closely agrees with the intuition, and captures cross hybridization between the probes

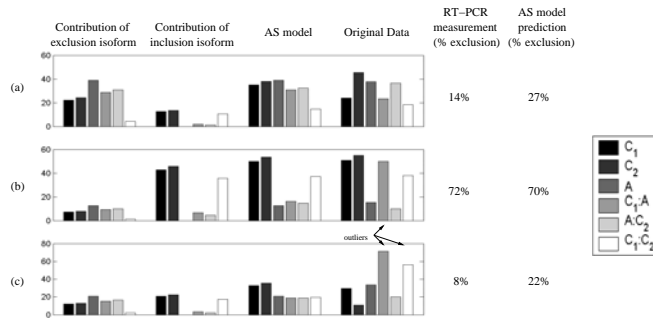

Figure 5: Three examples of data cases and their predictions. (a) The data does not follow our notion of single cassette exon AS, but the AS level is predicted accurately by the model.(b) The probe $C_1$:A is marked as outlier, allowing the model to predict the other probes accurately. (c) Two probes are marked as outliers, and the model is still successful in predicting the AS levels.

## 3 Making biological predictions about alternative splicing

The results presented in this paper were obtained using two stages of learning. In the first step, the weight matrix, $\Lambda$, is learned on a subset of the data that is selected for quality. Two selection criteria were used: (a) sequencing data was used to select those cases for which, with high confidence, no other AS event is present (Figure 1) and (b) probe sets were selected for high expression, as determined by a set of negative controls. The second selection criterion is motivated by the common assumption that low intensity measurements are of lesser quality (see Section 1.1). In the second step, $\Lambda$ is kept fixed, and we introduce the additional constraint that the noise is isotropic ($\Psi = \psi I$) and learn on the entire data set. The constraint on the noise is introduced to prevent the model from using only a subset of the six probes for making the final set of predictions.

We show a typical learned set of weights in Figure 4. The weights fit well with our intuition of what they should be to capture the presence of the two isoforms. Moreover, the learned weights account for the specific trends in the data. Examples of model prediction based on the microarray data are shown in Figure 5.

Due to the nature of the microarray data, we do not expect all the inferred abundances to be equally good, and we devised a scoring criterion that ranks each AS event based on its fit to the model. Intuitively, given two input vectors that are equivalent up to a scale factor, with inferred MAP estimations that are equal up to the same scale factor, we would like their scores to be identical. The scoring criterion used, therefore is $\sum_k (x_k - r\Lambda_k \mathbf{s})^2 / (x_k +$

| Rank | Pearson's correlation coefficient | False positive rate |
|---|---|---|
| 500 | 0.94 | 0.11 |
| 1000 | 0.95 | 0.08 |
| 2000 | 0.95 | 0.05 |
| 5000 | 0.79 | 0.2 |
| 10000 | 0.79 | 0.25 |
| 15000 | 0.78 | 0.29 |
| 20000 | 0.75 | 0.32 |
| 30000 | 0.65 | 0.42 |

Table 1: Model performance evaluated at various ranks. Using 180 RT-PCR measurements, we are able to predict the model's performance at various ranks. Two evaluation criteria are used: Pearson's correlation coefficient between the model's predictions and the RT-PCR measurements and false positive rate, where a prediction is considered to be false positive if it is more than 15% away from the RT-PCR measurement.

$r\Lambda_k\mathbf{s})^2$, where the MAP estimations for $r$ and $\mathbf{s}$ are used. This scoring criterion can be viewed as proportional to the sum of noise to signal ratios, as estimated using the two values given by the observation and the model's best prediction of that observation.

Since it is the relative amount of the isoforms that is of most interest, we need to use the inferred distribution of the isoform abundances to obtain an estimate for the relative levels of AS. It is not immediately clear how this should be done. We do, however, have RT-PCR measurements for 180 AS events to guide us (see figure 6 for details). Using the top 50 ranked RT-PCR measurement, we fit three parameters, $\{a_1, a_2, a_3\}$, such that the proportion of excluded isoform present, $p$, is given by $p = a_1 \frac{s_2}{s_1 + a_2 s_2} + a_3$, where $s_1$ is the MAP estimation of the abundance of the inclusion isoform, $s_2$ is the MAP estimation of the abundance of the exclusion isoform, and the RT-PCR measurement are used for target $p$. The parameters are fitted using gradient descent on a least squared error (LSE) evaluation criterion.

We used two criteria to evaluate the quality of the AS model predictions. Pearson's correlation coefficient (PCC) is used to evaluate the overall ability of the model to correctly estimate trends in the data. PCC is invariant to affine transformation and so is independent of the transformation parameters $a_1$ and $a_3$ discussed above, while the parameter $a_2$ was found to effect PCC very little. The PCC stays above 0.75 for the top two thirds ranked predictions. The second evaluation criterion used is the false positive rate, where a prediction is considered to be false positive if it is more than 15% away from the RT-PCR measurement. This allows us to say, for example, that if a prediction is within the top 10000, we are 75% confident that it is within 15% of the actual levels of AS.

## 4   Summary

We designed a novel AS model for the inference of the relative abundance of two alternatively spliced isoforms from six measurements. Unsupervised learning in the model is performed using a structured variational EM algorithm, which correctly captures the underlying structure of the data, as suggested by its biological nature. The AS model, though presented here for a cassette exon AS events, can be used to learn any type of AS, and with a simple adjustment, multiple types.

The predictions obtained from the AS model are currently being used to verify various claims about the role of AS in evolution and functional genomics, and to help identify sequences that affect the regulation of AS.

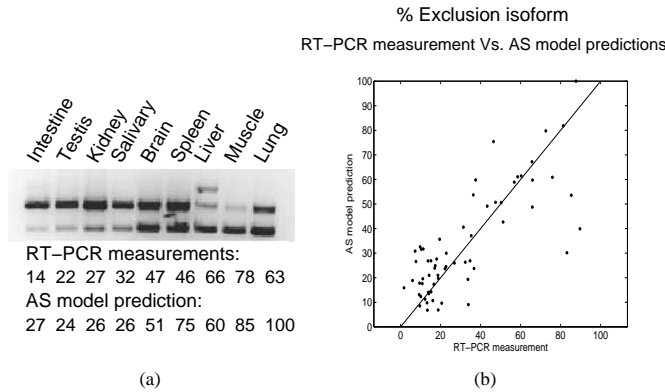

% Exclusion isoform
RT–PCR measurement Vs. AS model predictions

RT–PCR measurements:
14 22 27 32 47 46 66 78 63
AS model prediction:
27 24 26 26 51 75 60 85 100

(a)                                (b)

Figure 6: (a) Sample RT-PCR. RNA extracted from the cell is reverse-transcribed to DNA, amplified and labelled with radioactive or fluorescent molecules. The sample is pulled through a viscous gel in an electric field (DNA, being an acid, is positively charged). Shorter strands travel further through the gel than longer ones, resulting in two distinct bands, corresponding to the two isoforms, when exposed to a photosensitive or x-ray film. (b) A scatter plot showing the RT-PCR measurements as compared to the AS model predictions. The plot shows all available RT-PCR measurements with a rank of 8000 or better.

The AS model presented assumes a single weight matrix for all data cases. This is an oversimplified view of the data, and current work is being carried out in identifying probe specific expression profiles. However, due to the low dimensionality of the problem (10 tissues, six probes per event), care must be taken to avoid overfitting and to ensure meaningful interpretations.

## Acknowledgments

We would like to thank Wen Zhang, Naveed Mohammad, and Timothy Hughes for their contributions in generating the data set. This work was funded in part by an operating and infrastructure grants from the CIHR and CFI, and a operating grants from NSERC and a Premier's Research Excellence Award.

## References

[1] J. M. Johnson et al. Genome-wide survey of human alternative pre-mrna splicing with exon junction microarrays. *Science*, 302:2141–44, 2003.

[2] L. Cartegni et al. Listening to silence and understanding nonsense: exonic mutations that affect splicing. *Nature Gen. Rev.*, 3:285–98, 2002.

[3] Q. Pan et al. Revealing global regulatory features of mammalian alternative splicing using a quantitative microarray platform. *Molecular Cell*, 16(6):929–41, 2004.

[4] Q. Pan et al. Alternative splicing of conserved exons is frequently species specific in human and mouse. *Trends Gen.*, In Press, 2005.

[5] M. I. Jordan, Z. Ghahramani, T. Jaakkola, and Lawrence K. Saul. An introduction to variational methods for graphical models. *Machine Learning*, 37(2):183– 233, 1999.

[6] R. M. Neal and G. E. Hinton. A view of the em algorithm that justifies incremental, sparse, and other variants. In *Learning in Graphical Models*. Cambridge, MIT Press, 1998.

[7] D. M. Rocke and B. Durbin. A model for measurement error for gene expression arrays. *Journal of Computational Biology*, 8(6):557–69, 2001.
